# Block Variable Selection in Multivariate Regression and High-dimensional Causal Inference

**Aurélie C. Lozano, Vikas Sindhwani**
IBM T.J. Watson Research Center,
1101 Kitchawan Road,
Yorktown Heights NY 10598,USA
{aclozano,vsindhw}@us.ibm.com

## Abstract

We consider multivariate regression problems involving high-dimensional predictor and response spaces. To efficiently address such problems, we propose a variable selection method, Multivariate Group Orthogonal Matching Pursuit, which extends the standard Orthogonal Matching Pursuit technique. This extension accounts for arbitrary sparsity patterns induced by domain-specific groupings over both input and output variables, while also taking advantage of the correlation that may exist between the multiple outputs. Within this framework, we then formulate the problem of inferring causal relationships over a collection of high-dimensional time series variables. When applied to time-evolving social media content, our models yield a new family of causality-based influence measures that may be seen as an alternative to the classic PageRank algorithm traditionally applied to hyperlink graphs. Theoretical guarantees, extensive simulations and empirical studies confirm the generality and value of our framework.

## 1  Introduction

The broad goal of supervised learning is to effectively learn unknown functional dependencies between a set of input variables and a set of output variables, given a finite collection of training examples. This paper is at the intersection of two key topics that arise in this context.

The first topic is *Multivariate Regression* [4, 2, 24] which generalizes basic single-output regression to settings involving multiple output variables with potentially significant correlations between them. Applications of multivariate regression models include chemometrics, econometrics and computational biology. Multivariate Regression may be viewed as the classical precursor to many modern techniques in machine learning such as multi-task learning [15, 16, 1] and structured output prediction [18, 10, 22]. These techniques are output-centric in the sense that they attempt to exploit dependencies between output variables to learn joint models that generalize better than those that treat outputs independently.

The second topic is that of *sparsity* [3], *variable selection* and the broader notion of regularization [20]. The view here is input-centric in the following specific sense. In very high dimensional problems where the number of input variables may exceed the number of examples, the only hope for avoiding overfitting is via some form of aggressive capacity control over the family of dependencies being explored by the learning algorithm. This capacity control may be implemented in various ways, e.g., via dimensionality reduction, input variable selection or regularized risk minimization. Estimation of sparse models that are supported on a small set of input variables is a highly active and very successful strand of research in machine learning. It encompasses $l_1$ regularization (e.g., Lasso [19]) and matching pursuit techniques [13] which come with theoretical guarantees on the recovery of the exact support under certain conditions. Particularly pertinent to this paper is the

notion of *group sparsity*. In many problems involving very high-dimensional datasets, it is natural to enforce the prior knowledge that the support of the model should be a union over domain-specific groups of features. For instance, Group Lasso [23] extends Lasso, and Group-OMP [12, 9] extends matching pursuit techniques to this setting.

In view of these two topics, we consider here very high dimensional problems involving a large number of output variables. We address the problem of enforcing sparsity via variable selection in multivariate linear models where regularization becomes crucial since the number of parameters grows not only with the data dimensionality but also the number of outputs. Our approach is guided by the following desiderata: (a) performing variable selection for each output in isolation may be highly suboptimal since the input variables which are relevant to (a subset of) the outputs may only exhibit weak correlation with each individual output. It is also desirable to leverage information on the relatedness between outputs, so as to guide the decision on the relevance of a certain input variable to a certain output, using additional evidence based on the relevance to *related outputs*. (b) It is desirable to take into account any grouping structure that may exist between input and output variables. In the presence of noisy data, inclusion decisions made at the group level may be more robust than those at the level of individual variables.

To efficiently satisfy the above desiderata, we propose *Multivariate Group Orthogonal Matching Pursuit* (MGOMP) for enforcing arbitrary block sparsity patterns in multivariate regression coefficients. These patterns are specified by groups defined over both input and output variables. In particular, MGOMP can handle cases where the set of relevant features may differ from one response (group) to another, and is thus more general than simultaneous variable selection procedures (e.g. S-OMP of [21]), as simultaneity of the selection in MGOMP is enforced within groups of related output variables rather than the entire set of outputs. MGOMP also generalizes the Group-OMP algorithm of [12] to the multivariate regression case. We provide theoretical guarantees on the quality of the model in terms of correctness of group variable selection and regression coefficient estimation. We present empirical results on simulated datasets that illustrate the strength of our technique.

We then focus on applying MGOMP to *high-dimensional multivariate time series analysis* problems. Specifically, we propose a novel application of multivariate regression methods with variable selection, namely that of inferring key influencers in online social communities, a problem of increasing importance with the rise of planetary scale web 2.0 platforms such as Facebook, Twitter, and innumerable discussion forums and blog sites. We rigorously map this problem to that of inferring causal influence relationships. Using special cases of MGOMP, we extend the classical notion of Granger Causality [7] which provides an operational notion of causality in time series analysis, to apply to a collection of multivariate time series variables representing the evolving textual content of a community of bloggers. The sparsity structure of the resulting model induces a weighted causal graph that encodes influence relationships. While we use blog communities to concretize the application of our models, our ideas hold more generally to a wider class of spatio temporal causal modeling problems. In particular, our formulation gives rise to a new class of influence measures that we call *GrangerRanks*, that may be seen as causality-based alternatives to hyperlink-based ranking techniques like the PageRank [17], popularized by Google in the early days of the internet. Empirical results on a diverse collection of real-world key influencer problems clearly show the value of our models.

## 2  Variable Group Selection in Multivariate Regression

Let us begin by recalling the multivariate regression model, $\mathbf{Y} = \mathbf{X}\bar{\mathbf{A}} + \mathbf{E}$, where $\mathbf{Y} \in \mathbf{R}^{n \times K}$ is the output matrix formed by $n$ training examples on $K$ output variables, $\mathbf{X} \in \mathbf{R}^{n \times p}$ is the data matrix whose rows are $p$-dimensional feature vectors for the $n$ training examples, $\bar{\mathbf{A}}$ is the $p \times K$ matrix formed by the true regression coefficients one wishes to estimate, and $\mathbf{E}$ is the $n \times K$ error matrix. The row vectors of $\mathbf{E}$, are assumed to be independently sampled from $N(0, \mathbf{\Sigma})$ where $\mathbf{\Sigma}$ is the $K \times K$ error covariance matrix. For simplicity of notation we assume without loss of generality that the columns of $\mathbf{X}$ and $\mathbf{Y}$ have been centered so we need not deal with intercept terms.

The negative log-likelihood function (up to a constant) corresponding to the aforementioned model can be expressed as

$$-l(\mathbf{A}, \mathbf{\Sigma}) = \operatorname{tr}\left((\mathbf{Y} - \mathbf{X}\mathbf{A})^T(\mathbf{Y} - \mathbf{X}\mathbf{A})\mathbf{\Sigma}^{-1}\right) - n\log\left|\mathbf{\Sigma}^{-1}\right|, \tag{1}$$

where $\mathbf{A}$ is any estimate of $\bar{\mathbf{A}}$, and $|\cdot|$ denotes the determinant of a matrix. The maximum likelihood estimator is the Ordinary Least Squares (OLS) estimator $\hat{\mathbf{A}}^{\text{OLS}} = (\mathbf{X}^T\mathbf{X})^{-1}\mathbf{X}^T\mathbf{Y}$, namely, the concatenation of the OLS estimates for each of the $K$ outputs taken separately, irrespective of $\mathbf{\Sigma}$. This suggests its suboptimality as the relatedness of the responses is disregarded. Also the OLS estimator is known to perform poorly in the case of high dimensional predictors and/or when the predictors are highly correlated. To alleviate these issues, several methods have been proposed that are based on dimension reduction. Among those, variable selection methods are most popular as they lead to parsimonious and interpretable models, which is desirable in many applications. Clearly, however, variable selection in multiple output regression is particularly challenging in the presence of high dimensional feature vectors as well as possibly a large number of responses.

In many applications, including high-dimensional time series analysis and causal modeling settings showcased later in this paper, it is possible to provide domain specific guidance for variable selection by imposing a sparsity structure on $\mathbf{A}$. Let $\mathcal{I} = \{I_1 \ldots I_L\}$ denote the set formed by $L$ (possibly overlapping) groups of input variables where $I_k \subset \{1 \ldots p\}, k = 1, \ldots L$. Let $\mathcal{O} = \{O_1 \ldots O_M\}$ denote the set formed by $M$ (possibly overlapping) groups of output variables where $O_k \subset \{1 \ldots K\}, k = 1, \ldots, M$. Note that if certain variables do not belong to any group, they may be considered to be groups of size 1. These group definitions specify a block sparsity/support pattern on $\mathbf{A}$. Without loss of generality, we assume that column indices are permuted so that groups go over contiguous indices. We now outline a novel algorithm, *Multivariate Group Orthogonal Matching Pursuit* (MGOMP), that seeks to minimize the negative log-likelihood associated with the multivariate regression model subject to the constraint that the support (set of non-zeros) of the regression coefficient matrix, $\mathbf{A}$, is a union of blocks formed by input and output variable groupings[1].

## 2.1 Multivariate Group Orthogonal Matching Pursuit

The MGOMP procedure performs greedy pursuit with respect to the loss function

$$L_C(\mathbf{A}) = \text{tr}\left((\mathbf{Y} - \mathbf{XA})^T(\mathbf{Y} - \mathbf{XA})\mathbf{C}\right), \tag{2}$$

where $\mathbf{C}$ is an estimate of the precision matrix $\mathbf{\Sigma}^{-1}$, given as input. Possible estimates include the sample estimate using residual error obtained from running univariate Group-OMP for each response individually. In addition to leveraging the grouping information via block sparsity constraints, MGOMP is able to incorporate additional information on the relatedness among output variables as implicitly encoded in the error covariance matrix $\mathbf{\Sigma}$, noting that the latter is also the covariance matrix of the response $\mathbf{Y}$ conditioned on the predictor matrix $\mathbf{X}$. Existing variable selection methods often ignore this information and deal instead with (regularized versions of) the simplified objective $\text{tr}\left((\mathbf{Y} - \mathbf{XA})^T(\mathbf{Y} - \mathbf{XA})\right)$, thereby implicitly assuming that $\mathbf{\Sigma} = \mathbf{I}$.

Before outlining the details of MGOMP, we first need to introduce some notation. For any set of output variables $O \subset \{1, \ldots, K\}$, denote by $C_O$ the restriction of the $K \times K$ precision matrix $C$ to columns corresponding to the output variables in $O$, and by $C_{O,O}$ similar restriction to both columns and rows. For any set of input variables $I \subset \{1, \ldots, p\}$, denote by $\mathbf{X}_I$ the restriction of $\mathbf{X}$ to columns corresponding to the input variables in $I$. Furthermore, to simplify the exposition, we assume in the remainder of the paper that for each group of input variables $I_s \in \mathcal{I}$, $\mathbf{X}_{I_s}$ is orthonormalized, i.e., $\mathbf{X}_{I_s}^T\mathbf{X}_{I_s} = \mathbf{I}$. Denote by $\mathbf{A}^{(m)}$ the estimate of the regression coefficient matrix at iteration $m$, and by $\mathbf{R}^{(m)}$ the corresponding matrix of residuals, i.e. $\mathbf{R}^{(m)} = \mathbf{Y} - \mathbf{XA}^{(m)}$.

The MGOMP procedure iterates between two steps : (a) Block Variable Selection and (b) Coefficient matrix re-estimation with selected block. We now outline the details of these two steps.

**Block Variable Selection**: In this step, each block, $(I_r, O_s)$, is evaluated with respect to how much its introduction into $\mathbf{A}^{m-1}$ can reduce residual loss. Namely, at round $m$, the procedure selects the block $(I_r, O_s)$ that minimizes

$$\underset{1 \le r \le L, 1 \le s \le M}{\arg\min} \ \underset{\mathbf{A}:A_{v,w}=0, v \notin I_r, w \notin O_s}{\min} \left(L_C(\mathbf{A}^{(m-1)} + \mathbf{A}) - L_C(\mathbf{A}^{(m-1)})\right).$$

Note that when the minimum attained falls below $\epsilon$, the algorithm is stopped. Using standard Linear Algebra, the block variable selection criteria simplifies to

$$(r^{(m)}, s^{(m)}) = \arg\max_{r,s} \text{tr}\left((\mathbf{X}_{I_r}^T \mathbf{R}^{(m-1)} C_{O_s})^T (\mathbf{X}_{I_r}^T \mathbf{R}^{(m-1)} C_{O_s})(C_{O_s,O_s}^{-1})\right). \qquad (3)$$

From the above equation, it is clear that the relatedness between output variables is taken into account in the block selection process.

**Coefficient Re-estimation:**

Let $\mathcal{M}^{(m-1)}$ be the set of blocks selected up to iteration $m-1$ . The set is now updated to include the selected block of variables $(I_{r^{(m)}}, O_{s^{(m)}})$, i.e., $\mathcal{M}^{(m)} = \mathcal{M}^{(m-1)} \cup \{(I_{r^{(m)}}, O_{s^{(m)}})\}$. The regression coefficient matrix is then re-estimated as $\mathbf{A}^{(m)} = \hat{\mathbf{A}}_{\mathbf{X}}(\mathcal{M}^{(m)}, \mathbf{Y})$, where

$$\hat{\mathbf{A}}_{\mathbf{X}}(\mathcal{M}^{(m)}, \mathbf{Y}) = \arg\min_{\mathbf{A} \in \mathbb{R}^{p \times K}} L_C(\mathbf{A}) \text{ subject to } \text{supp}(\mathbf{A}) \subseteq \mathcal{M}^{(m)}. \qquad (4)$$

Since certain features are only relevant to a subset of responses, here the precision matrix estimate $\mathbf{C}$ comes into play, and the problem can not be decoupled. However, a closed form solution for (4) can be derived by recalling the following matrix identities [8],

$$\text{tr}(\mathbf{M}_1^T \mathbf{M}_2 \mathbf{M}_3 \mathbf{M}_4^T) = vec(\mathbf{M}_1)^T (\mathbf{M}_4 \otimes \mathbf{M}_2) vec(\mathbf{M}_3), \qquad (5)$$
$$vec(\mathbf{M}_1 \mathbf{M}_2) = (\mathbf{I} \otimes \mathbf{M}_1) vec(\mathbf{M}_2), \qquad (6)$$

where $vec$ denotes the matrix vectorization, $\otimes$ the Kronecker product, and $\mathbf{I}$ the identity matrix. From (5), we have

$$\text{tr}\left((\mathbf{Y} - \mathbf{XA})^T (\mathbf{Y} - \mathbf{XA})\mathbf{C}\right) = (vec(\mathbf{Y} - \mathbf{XA}))^T (\mathbf{C} \otimes \mathbf{I}_n)(vec(\mathbf{Y} - \mathbf{XA})). \qquad (7)$$

For a set of selected blocks, say $\mathcal{M}$, denote by $O(\mathcal{M})$ the union of the output groups in $\mathcal{M}$. Let $\tilde{\mathbf{C}} = \mathbf{C}_{O(\mathcal{M}),O(\mathcal{M})} \otimes \mathbf{I}_n$ and $\tilde{Y} = vec(\mathbf{Y}_{O(\mathcal{M})})$. For each output group $O_s$ in $\mathcal{M}$, let $I(O_s) = \cup_{(I_r,O_s) \in \mathcal{M}} I_r$. Finally define $\tilde{\mathbf{X}}$ such that $\tilde{\mathbf{X}} = \text{diag}\left\{\mathbf{I}_{|O_s|} \otimes \mathbf{X}_{I(O_s)}, O_s \in O(\mathcal{M})\right\}$. Using (7) and (6) one can show that the non-zero entries of $vec(\hat{\mathbf{A}}_{\mathbf{X}}(\mathcal{M}, \mathbf{Y}))$, namely those corresponding to the support induced by $\mathcal{M}$, are given by $\hat{\alpha} = \left(\tilde{\mathbf{X}}^T \tilde{\mathbf{C}} \tilde{\mathbf{X}}\right)^{-1} \left(\tilde{\mathbf{X}}^T \tilde{\mathbf{C}}\right) \tilde{Y}$, thus providing a closed-form formula for the coefficient re-estimation step.

To conclude this section, we note that we could also consider preforming alternate optimization of the objective in (1) over $\mathbf{A}$ and $\Sigma$, using *MGOMP* to optimize over $\mathbf{A}$ for a fixed estimate of $\Sigma$, and using a covariance estimation algorithm (e.g. Graphical Lasso [5]) to estimate $\Sigma$ with fixed $\mathbf{A}$.

### 2.2 Theoretical Performance Guarantees for *MGOMP*

In this section we show that under certain conditions *MGOMP* can identify the correct blocks of variables and provide an upperbound on the maximum absolute difference between the estimated and true regression coefficients. We assume that the estimate of the error precision matrix, $\mathbf{C}$, is in agreement with the specification of the output groups, namely that $\mathbf{C}_{i,j} = 0$ if $i$ and $j$ belong to different output groups.

For each output variable group $O_k$, denote by $\mathcal{G}_{\text{good}}(k)$ the set formed by the input groups included in the true model for the regressions in $O_k$, and let $\mathcal{G}_{\text{bad}}(k)$ be the set formed by all the pairs that are not included. Similarly denote by $\mathcal{M}_{\text{good}}$ the set formed by the pairs of input and output variable groups included in the true model, and $\mathcal{M}_{\text{bad}}$ be the set formed by all the pairs that are not included.

Before we can state the theorem, we need to define the parameters that are key in the conditions for consistency. Let $\rho_X(\mathcal{M}_{\text{good}}) = \min_{k \in \{1,\dots,M\}} \inf_\alpha \left\{\|\mathbf{X}\alpha\|_2^2 / \|\alpha\|_2^2 : \text{supp}(\alpha) \subseteq \mathcal{G}_{\text{good}}(k)\right\}$, namely $\rho_X(\mathcal{M}_{\text{good}})$ is the minimum over the output groups $O_k$ of the smallest eigenvalue of $\mathbf{X}_{\mathcal{G}_{\text{good}}(k)}^T \mathbf{X}_{\mathcal{G}_{\text{good}}(k)}$.

For each output group $O_k$, define generally for any $u = \{u_1, \dots, u_{|\mathcal{G}_{\text{good}}(k)|}\}$ and $v = \{v_1, \dots, v_{|\mathcal{G}_{\text{bad}}(k)|}\}$,

$$\|u\|_{(2,1)}^{\text{good}(k)} = \sum_{G_i \in \mathcal{G}_{\text{good}}(k)} \sqrt{\sum_{j \in G_i} u_j^2}, \text{ and } \|v\|_{(2,1)}^{\text{bad}(k)} = \sum_{G_i \in \mathcal{G}_{\text{bad}}(k)} \sqrt{\sum_{j \in G_i} v_j^2}.$$

For any matrix $\mathbf{M} \in \mathbf{R}^{|\mathcal{G}_{\text{good}}(k)| \times |\mathcal{G}_{\text{bad}}(k)|}$, let $\|\mathbf{M}\|_{(2,1)}^{\text{good/bad(k)}} = \sup_{\|v\|_{(2,1)}^{\text{bad(k)}}=1} \|\mathbf{M}v\|_{(2,1)}^{\text{good(k)}}$.

Then we define $\mu_X(\mathcal{M}_{\text{good}}) = \max_{k \in \{1,\dots,M\}} \|\mathbf{X}_{\mathcal{G}_{\text{good}}(k)}^{+} \mathbf{X}_{\mathcal{G}_{\text{bad}}(k)}\|_{(2,1)}^{\text{good/bad(k)}}$, where $\mathbf{X}^{+}$ denotes the Moore-Penrose pseudoinverse of $\mathbf{X}$. We are now able to state the consistency theorem.

**Theorem 1.** *Assume that* $\mu_X(\mathcal{M}_{\text{good}}) < 1$ *and* $0 < \rho_X(\mathcal{M}_{\text{good}}) \leq 1$. *For any* $\eta \in (0,1/2)$, *with probability at least* $1 - 2\eta$, *if the stopping criterion of MGOMP is such that* $\epsilon > \frac{1}{1-\mu_X(\mathcal{M}_{\text{good}})}\sqrt{2pK\ln(2pK/\eta)}$ *and* $\min_{k\in\{1,\dots,M\},I_j\in\mathcal{G}_{\text{good}}(k)}\|\bar{\mathbf{A}}_{I_j,O_k}\|_F \geq \sqrt{8}\epsilon\rho_X(\mathcal{M}_{\text{good}})^{-1}$ *then when the algorithm stops* $\mathcal{M}^{(m-1)} = \mathcal{M}_{\text{good}}$ *and* $\|\mathbf{A}^{(m-1)} - \bar{\mathbf{A}}\|_{\max} \leq \sqrt{(2\ln(2|\mathcal{M}_{\text{good}}|/\eta))/\rho_X(\mathcal{M}_{\text{good}})}$.

*Proof.* The multivariate regression model $\mathbf{Y} = \mathbf{X}\bar{\mathbf{A}} + \mathbf{E}$ can be rewritten in an equivalent univariate form with white noise: $\tilde{Y} = (\mathbf{I_K} \otimes \mathbf{X})\bar{\alpha} + \eta$, where $\bar{\alpha} = \text{vec}(\bar{\mathbf{A}})$, $\tilde{Y} = \text{diag}\left\{\frac{1}{C_{k,k}^{1/2}}\mathbf{I}_n\right\}_{k=1}^{K} \text{vec}(\mathbf{Y}\mathbf{C}^{1/2})$, and $\eta$ is formed by i.i.d samples from $N(0,1)$. We can see that applying the MGOMP procedure is equivalent to applying the Group-OMP procedure [12] to the above vectorized regression model, using as grouping structure that naturally induced by the input-output groups originally considered for MGOMP. The theorem then follows from Theorem 3 in [12] and translating the univariate conditions for consistency into their multivariate counterparts via $\mu_X(\mathcal{M}_{\text{good}})$ and $\rho_X(\mathcal{M}_{\text{good}})$. Since $C$ is such that $C_{i,j} = 0$ for any $i,j$ belonging to distinct groups, the entries in $\tilde{Y}$ do not mix components of $\mathbf{Y}$ from different output groups and hence the error covariance matrix does not appear in the consistency conditions. $\square$

Note that the theorem can also be re-stated with an alternative condition on the amplitude of the true regression coefficient: $\min_{k\in\{1,\dots,M\},I_j\in\mathcal{G}_{\text{good}}(k)}\min_{s\in O_k}\|\bar{\mathbf{A}}_{I_j,k}\|_2 \geq \sqrt{8}\epsilon\rho_X(\mathcal{M}_{\text{good}})^{-1}/\sqrt{|O_k|}$ which suggests that the amplitude of the true regression coefficients is allowed to be smaller in MGOMP compared to Group-OMP on individual regressions. Intuitively, through MGOMP we are combining information from multiple regressions, thus improving our capability to identify the correct groups.

## 2.3 Simulation Results

We empirically evaluate the performance of our method against representative variable selection methods, in terms of accuracy of prediction and variable (group) selection. As a measure of variable selection accuracy we use the $F_1$ measure, which is defined as $F_1 = \frac{2PR}{P+R}$, where $P$ denotes the precision and $R$ denotes the recall. To compute the variable group $F_1$ of a variable selection method, we consider a group to be selected if *any* of the variables in the group is selected. As a measure of prediction accuracy we use the average squared error on a test set. For all the greedy pursuit methods, we consider the "holdout validated" estimates. Namely, we select the iteration number that minimizes the average squared error on a validation set. For univariate methods, we consider individual selection of the iteration number for each univariate regression (joint selection of a common iteration number across the univariate regressions led to worse results in the setting considered). For each setting, we ran 50 runs, each with 50 observations for training, 50 for validation and 50 for testing.

We consider an $n \times p$ predictor matrix $\mathbf{X}$, where the rows are generated independently according to $N_p(0,\mathbf{S})$, with $\mathbf{S}_{i,j} = 0.7^{|i-j|}$. The $n \times K$ error matrix $\mathbf{E}$ is generated according to $N_K(0,\boldsymbol{\Sigma})$, with $\boldsymbol{\Sigma}_{i,j} = \rho^{|i-j|}$, where $\rho \in \{0, 0.5, 0.7, 0/9\}$. We consider a model with 3rd order polynomial expansion: $[\mathbf{Y}_{T_1},\dots,\mathbf{Y}_{T_M}] = \mathbf{X}[\mathbf{A}_{1,T_1},\dots,\mathbf{A}_{1,T_M}] + \mathbf{X^2}[\mathbf{A}_{2,T_1},\dots,\mathbf{A}_{2,T_M}] + \mathbf{X^3}[\mathbf{A}_{3,T_1},\dots,\mathbf{A}_{3,T_M}] + \mathbf{E}$. Here we abuse notation to denote by $\mathbf{X}^q$ the matrix such that $\mathbf{X}_{i,j}^q = (\mathbf{X}_{i,j})^q$. $T_1,\dots,T_M$ are the target groups. For each $k$, each row of $[\mathbf{A}_{1,T_k},\dots,\mathbf{A}_{3,T_k}]$ is either all non-zero or all zero, according to Bernoulli draws with success probability 0.1. Then for each non-zero entry of $\mathbf{A}_{i,T_k}$, independently, we set its value according to $N(0,1)$. The number of features for $X$ is set to 20. Hence we consider 60 variables grouped into 20 groups corresponding the the 3rd degree polynomial expansion. The number of regressions is set to 60. We consider 20 regression groups $(T_1,\dots T_{20})$, each of size 3.

| Parallel runs | $(\mathbf{p}, \mathbf{L})$ | $(\mathbf{K}, \mathbf{M})$ | **Precision matrix estimate** | **Method** |
|---|---|---|---|---|
| $K$ | $(p, p)$ | $(1, 1)$ | Not applicable | OMP [13] |
| $K$ | $(p, L)$ | $(1, 1)$ | Not applicable | Group-OMP [12] |
| $1$ | $(p, p)$ | $(K, 1)$ | Identity matrix | S-OMP [21] |
| $1$ | $(p, L)$ | $(K, M)$ | Identity matrix | MGOMP(Id) |
| $1$ | $(p, L)$ | $(K, M)$ | Estimate from univariate OMP fits | MGOMP(C) |
| $M$ | $(p, L)$ | $(M', 1)$ | Identity matrix | MGOMP(Parallel) |

Table 1: Various matching pursuit methods and their corresponding parameters.

| $\rho$ | MGOMP (C) | MGOMP (Id) | MGOMP(Parallel) | Group-OMP | OMP |
|---|---|---|---|---|---|
| 0.9 | $\mathbf{0.863 \pm 0.003}$ | $0.818 \pm 0.003$ | $0.762 \pm 0.003$ | $0.646 \pm 0.007$ | $0.517 \pm 0.006$ |
| 0.7 | $\mathbf{0.850 \pm 0.002}$ | $0.806 \pm 0.003$ | $0.757 \pm 0.003$ | $0.631 \pm 0.008$ | $0.517 \pm 0.007$ |
| 0.5 | $\mathbf{0.850 \pm 0.003}$ | $0.802 \pm 0.004$ | $0.766 \pm 0.004$ | $0.641 \pm 0.006$ | $0.525 \pm 0.007$ |
| 0 | $0.847 \pm 0.004$ | $\mathbf{0.848 \pm 0.004}$ | $0.783 \pm 0.004$ | $0.651 \pm 0.007$ | $0.525 \pm 0.007$ |
| $\rho$ | MGOMP (C) | MGOMP (Id) | MGOMP(Parallel) | Group-OMP | OMP |
| 0.9 | $\mathbf{3.009 \pm 0.234}$ | $3.324 \pm 0.273$ | $4.086 \pm 0.169$ | $6.165 \pm 0.317$ | $6.978 \pm 0.206$ |
| 0.7 | $\mathbf{3.114 \pm 0.252}$ | $3.555 \pm 0.287$ | $4.461 \pm 0.159$ | $8.170 \pm 0.328$ | $8.14 \pm 0.390$ |
| 0.5 | $\mathbf{3.117 \pm 0.234}$ | $3.630 \pm 0.281$ | $4.499 \pm 0.288$ | $7.305 \pm 0.331$ | $8.098 \pm 0.323$ |
| 0 | $3.124 \pm 0.256$ | $\mathbf{3.123 \pm 0.262}$ | $3.852 \pm 0.185$ | $6.137 \pm 0.330$ | $7.414 \pm 0.331$ |

Table 2: Average $F_1$ score (top) and average test set squared error (bottom) for the models output by variants of MGOMP, Group-OMP and OMP under the settings of Table 1.

A dictionary of various matching pursuit methods and their corresponding parameters is provided in Table 1. In the table, note that MGOMP(Parallel) consists in running MGOMP separately for each regression group and $\mathbf{C}$ set to identity (Using $\mathbf{C}$ estimated from univariate OMP fits has negligible impact on performance and hence is omitted for conciseness.). The results are presented in Table 2.

Overall, in all the settings considered, MGOMP is superior both in terms of prediction and variable selection accuracy, and more so when the correlation between responses increases. Note that MGOMP is stable with respect to the choice of the precision matrix estimate. Indeed the advantage of MGOMP persists under imperfect estimates (Identity and sample estimate from univariate OMP fits) and varying degrees of error correlation. In addition, model selection appears to be more robust for MGOMP, which has only one stopping point (MGOMP has one path interleaving input variables for various regressions, while GOMP and OMP have $K$ paths, one path per univariate regression).

## 3 Granger Causality with Block Sparsity in Vector Autoregressive Models

### 3.1 Model Formulation

We begin by motivating our main application. The emergence of the web2.0 phenomenon has set in place a planetary-scale infrastructure for rapid proliferation of information and ideas. Social media platforms such as blogs, twitter accounts and online discussion sites are large-scale forums where every individual can voice a potentially influential public opinion. This unprecedented scale of unstructured user-generated web content presents new challenges to both consumers and companies alike. Which blogs or twitter accounts should a consumer follow in order to get a gist of the community opinion as a whole? How can a company identify bloggers whose commentary can change brand perceptions across this universe, so that marketing interventions can be effectively strategized? The problem of finding key influencers and authorities in online communities is central to any viable information triage solution, and is therefore attracting increasing attention [14, 6]. A traditional approach to this problem would treat it no different from the problem of ranking web-pages in a hyperlinked environment. Seminal ideas such as the PageRank [17] and Hubs-and-Authorities [11] were developed in this context, and in fact even celebrated as bringing a semblance of order to the web. However, the mechanics of opinion exchange and adoption makes the problem of inferring authority and influence in social media settings somewhat different from the problem of ranking generic web-pages. Consider the following example that typifies the process of opinion adoption. A consumer is looking to buy a laptop. She initiates a web search for the laptop model and browses several discussion and blog sites where that model has been reviewed. The reviews bring to her attention that among other nice features, the laptop also has excellent speaker quality. Next she buys the laptop and in a few days herself blogs about it. Arguably, conditional on being made aware of

speaker quality in the reviews she had read, she is more likely to herself comment on that aspect without necessarily attempting to find those sites again in order to link to them in her blog. In other words, the actual post content is the only trace that the opinion was implicitly absorbed. Moreover, the temporal order of events in this interaction is indicative of the *direction of causal influence*.

We formulate these intuitions rigorously in terms of the notion of *Granger Causality* [7] and then employ MGOMP for its implementation. For scalability, we work with MGOMP (Parallel), see table 1. Introduced by the Nobel prize winning economist, Clive Granger, this notion has proven useful as an operational notion of causality in time series analysis. It is based on the intuition that a cause should necessarily precede its effect, and in particular if a time series variable $X$ causally affects another $Y$, then the past values of $X$ should be helpful in predicting the future values of $Y$, beyond what can be predicted based on the past values of $Y$ alone.

Let $B_1 \ldots B_G$ denote a community of $G$ bloggers. With each blogger, we associate *content variables*, which consist of frequencies of words relevant to a topic across time. Specifically, given a dictionary of $K$ words and the time-stamp of each blog post, we record $w_i^{k,t}$, the frequency of the $k$th word for blogger $B_i$ at time $t$. Then, the *content* of blogger $B_i$ at time $t$ can be represented as $\mathbf{B}_i^t = [w_i^{1,t}, \ldots, w_i^{K,t}]$. The input to our model is a collection of multivariate time series, $\{\mathbf{B}_i^t\}_{t=1}^T$ $(1 \leq i \leq G)$, where $T$ is the timespan of our analysis. Our key intuition is that authorities and influencers are *causal drivers* of *future* discussions and opinions in the community. This may be phrased in the following terms:

**Granger Causality:** *A collection of bloggers is said to influence Blogger $B_i$ if their collective past content (blog posts) is predictive of the future content of Blogger $B_i$, with statistical significance, and more so than the past content of Blogger $B_i$ alone.*

The influence problem can thus be mapped to a variable group selection problem in a vector autoregressive model, i.e., in multivariate regression with $G \times K$ responses $\{\mathbf{B}_j^t, j = 1, 2 \ldots G\}$ in terms of variable groups $\{\{\mathbf{B}_j^{t-l}\}_{l=1}^d, j = 1, 2 \ldots G\}$: $[\mathbf{B}_1^t, \ldots, \mathbf{B}_G^t] = [\mathbf{B}_1^{t-1}, \ldots, \mathbf{B}_1^{t-d}, \ldots, \mathbf{B}_G^{t-1}, \ldots, \mathbf{B}_G^{t-d}]\mathbf{A} + \mathbf{E}$. We can then conclude that a certain blogger $B_i$ influences blogger $B_j$, if the variable group $\{\mathbf{B}_i^{t-l}\}_{l \in \{1, \ldots, d\}}$ is selected by the variable selection method for the responses concerning blogger $B_j$. For each blogger $B_j$, this can be viewed as an application of a Granger test on $B_j$ against bloggers $B_1, B_2, \ldots, B_G$. This induces a directed weighted graph over bloggers, which we call *causal graph,* where edge weights are derived from the underlying regression coefficients. We refer to influence measures on causal graphs as *GrangerRanks*. For example, GrangerPageRank refers to applying pagerank on the causal graph while GrangerOutDegree refers to computing out-degrees of nodes as a measure of causal influence.

## 3.2 Application: Causal Influence in Online Social Communities

**Proof of concept: Key Influencers in Theoretical Physics**: Drawn from a KDD Cup 2003 task, this dataset is publically available at: `http://www.cs.cornell.edu/projects/kddcup/datasets.html`. It consists of the latex sources of all papers in the *hep-th* portion of the arXiv (`http://arxiv.org`) In consultation with a theoretical physicist we did our analysis at a time granularity of 1 month. In total, the data spans 137 months. We created document term matrices using standard text processing techniques, over a vocabulary of 463 words chosen by running an unsupervised topic model. For each of the 9200 authors, we created a word-time matrix of size 463x137, which is the usage of the topic-specific key words across time. We considered one year, i.e., $d = 12$ months as maximum time lag. Our model produces the causal graph shown in Figure 1 showing influence relationships amongst high energy physicists. The table on the right side of Figure 1 lists the top 20 authors according to GrangerOutDegree (also marked on the graph), GrangerPagerRank and Citation Count. The model correctly identifies several leading figures such as Edward Witten, Cumrun Vafa as authorities in theoretical physics. In this domain, number of citations is commonly viewed as a valid measure of authority given disciplined scholarly practice of citing prior related work. Thus, we consider citation-count based ranking as the "ground truth". We also find that GrangerPageRank and GrangerOutDegree have high positive rank correlation with citation counts (0.728 and 0.384 respectively). This experiment confirms that our model agrees with how this community recognizes its authorities.

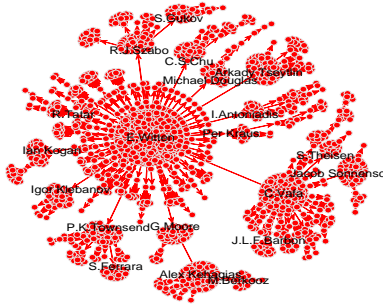

| GrangerOutdegree | GrangerPageRank | Citation Count |
| --- | --- | --- |
| E.Witten | E.Witten | E.Witten |
| C.Vafa | C.Vafa | N.Seiberg |
| Alex Kehagias | Alex Kehagias | C.Vafa |
| Arkady Tseytlin | Arkady Tseytlin | J.M.Maldacena |
| P.K.Townsend | P.K.Townsend | A.A.Sen |
| Jacob Sonnenschein | Jacob Sonnenschein | Andrew Strominger |
| Igor Klebanov | R.J.Szabo | Igor Klebanov |
| R.J.Szabo | G.Moore | Michael Douglas |
| G.Moore | Igor Klebanov | Arkady Tseytlin |
| Michael Douglas | Ian Kogan | L.Susskind |

Figure 1: Causal Graph and top authors in High-Energy Physics according to various measures.

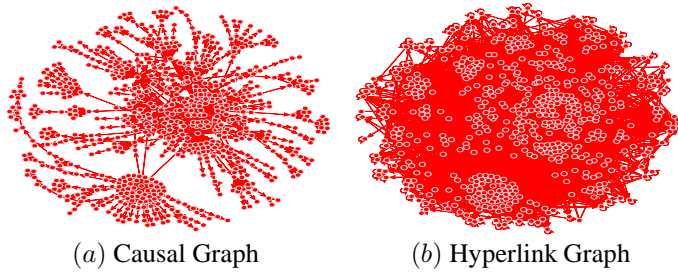

(*a*) Causal Graph          (*b*) Hyperlink Graph

Figure 2: Causal and hyperlink graphs for the lotus blog dataset.

**Real application: IBM Lotus Bloggers**: We crawled blogs pertaining to the IBM Lotus software brand. Our crawl process ran in conjunction with a relevance classifier that continuously filtered out posts irrelevant to Lotus discussions. Due to lack of space we omit preprocessing details that are similar to the previous application. In all, this dataset represents a Lotus blogging community of 684 bloggers, each associated with multiple time series describing the frequency of 96 words over a time period of 376 days. We considered one week i.e., $d = 7$ days as maximum time lag in this application. Figure 2 shows the causal graph learnt by our models on the left, and the hyperlink graph on the right. We notice that the causal graph is sparser than the hyperlink graph. By identifying the most significant causal relationships between bloggers, our causal graphs allow clearer inspection of the authorities and also appear to better expose striking sub-community structures in this blog community. We also computed the correlation between PageRank and Outdegrees computed over our causal graph and the hyperlink graph (0.44 and 0.65 respectively). We observe positive correlations indicating that measures computed on either graph partially capture related latent rankings, but at the same time are also sufficiently different from each other. Our results were also validated by domain experts.

## 4   Conclusion and Perspectives

We have provided a framework for learning sparse multivariate regression models, where the sparsity structure is induced by groupings defined over both input and output variables. We have shown that extended notions of Granger Causality for causal inference over high-dimensional time series can naturally be cast in this framework. This allows us to develop a causality-based perspective on the problem of identifying key influencers in online communities, leading to a new family of influence measures called GrangerRanks. We list several directions of interest for future work: optimizing time-lag selection; considering hierarchical group selection to identify pertinent causal relationships not only between bloggers but also between communities of bloggers; incorporating the hyperlink graph in the causal modeling; adapting our approach to produce topic specific rankings; developing online learning versions; and conducting further empirical studies on the properties of the causal graph in various applications of multivariate regression.

### Acknowledgments

We would like to thank Naoki Abe, Rick Lawrence, Estepan Meliksetian, Prem Melville and Grzegorz Swirszcz for their contributions to this work in a variety of ways.

## Footnotes

[1]We note that we could easily generalize this setting and MGOMP to deal with the more general case where there may be a different grouping structure for each output group, namely for each $O_k$, we could consider a different set $\mathcal{I}_{O_k}$ of input variable groups.

# References

[1] Andreas Argyriou, Theodoros Evgeniou, and Massimiliano Pontil. Convex multi-task feature learning. *Machine Learning*, 73(3):243–272, 2008.

[2] Leo Breiman and Jerome H Friedman. Predicting multivariate responses in multiple linear regression. *Journal of the Royal Statistical Society: Series B*, (1):1369–7412, 1997.

[3] M. Elad. Sparse and Redundant Representations: From Theory to Applications in Signal and Image Processing. Springer,2010

[4] Ildiko E. Frank and Jerome H. Friedman. A statistical view of some chemometrics regression tools. *Technometrics*, 35(2):109–135, 1993.

[5] J. Friedman, T. Hastie, and R. Tibshirani. Sparse inverse covariance estimation with the graphical lasso. *Biostatistics*, 9(3):432–441, July 2008.

[6] M. Gomez-Rodriguez and J. Leskovec and A. Krause. Inferring Networks of Diffusion and Influence, KDD 2010.

[7] C. Granger. Testing for causality: A personal viewpoint. *Journal of Economic Dynamics and Control*, 2:329–352, 1980.

[8] D. Harville. *Matrix Algebra from a Statistician's Perspective*. Springer, 1997.

[9] J. Huang, T. Zhang, and D. Metaxas D. Learning with structured sparsity, ICML 2009.

[10] T. Joachims. Structured output prediction with support vector machines. In *Joint IAPR International Workshops on Structural and Syntactic Pattern Recognition (SSPR) and Statistical Techniques in Pattern Recognition (SPR)*, pages 1–7, 2006.

[11] Jon M. Kleinberg. Authoritative sources in a hyperlinked environment. *Journal of the ACM*, 46:668–677, 1999.

[12] A.C. Lozano, G. Swirszcz, and N. Abe. Grouped orthogonal matching pursuit for variable selection and prediction. *Advances in Neural Information Processing Systems 22*, 2009.

[13] S. Mallat and Z. Zhang. Matching pursuits with time-frequency dictionaries. *IEEE Transactions on Signal Processing*, 1993.

[14] P. Melville, K. Subbian, C. Perlich, R. Lawrence and E. Meliksetian. A Predictive Perspective on Measures of Influence in Networks Workshop on Information in Networks (WIN-10), New York, September, 2010.

[15] Charles A. Micchelli and Massimiliano Pontil. Kernels for multi–task learning. In *NIPS*, 2004.

[16] G. Obozinski, B. Taskar, and M. Jordan. Multi-task feature selection. *Technical report*, 2006.

[17] L. Page, S. Brin, R. Motwani, and T. Winograd. The pagerank citation ranking: Bringing order to the web. *Technical Report, Stanford Digital Libraries*, 1998.

[18] Elisa Ricci, Tijl De Bie, and Nello Cristianini. Magic moments for structured output prediction. *Journal of Machine Learning Research*, 9:2803–2846, December 2008.

[19] R. Tibshirani. Regression shrinkage and selection via the lasso. *Journal of the Royal Statistical Society, Series B*, 58:267–288, 1994.

[20] A.N. Tikhonov. Regularization of incorrectly posed problems. *Sov. Math. Dokl*, 4:16241627, 1963.

[21] J.A. Tropp, A.C. Gilbert, and M.J. Strauss. Algorithms for simultaneous sparse approximation: part i: Greedy pursuit. *Sig. Proc.*, 86(3):572–588, 2006.

[22] I. Tsochantaridis, T. Hofmann, T. Joachims, and Y. Altun. Support vector machine learning for interdependent and structured output spaces. In *International Conference on Machine Learning (ICML)*, pages 104–112, 2004.

[23] M. Yuan and Y. Lin. Model selection and estimation in regression with grouped variables. *Journal of the Royal Statistical Society, Series B*, 68:49–67, 2006.

[24] Ming Yuan, Ali Ekici, Zhaosong Lu, and Renato Monteiro. Dimension reduction and coefficient estimation in multivariate linear regression. *Journal Of The Royal Statistical Society Series B*, 69(3):329–346, 2007.

